# Image Retrieval and Classification Using Local Distance Functions

**Andrea Frome**
Department of Computer Science
UC Berkeley
Berkeley, CA 94720
andrea.frome@gmail.com

**Yoram Singer**
Google, Inc.
Mountain View, CA 94043
singer@google.com

**Jitendra Malik**
Department of Computer Science
UC Berkeley
malik@cs.berkeley.edu

## Abstract

In this paper we introduce and experiment with a framework for learning local perceptual distance functions for visual recognition. We learn a distance function for *each training image* as a combination of elementary distances between patch-based visual features. We apply these combined local distance functions to the tasks of image retrieval and classification of novel images. On the Caltech 101 object recognition benchmark, we achieve 60.3% mean recognition across classes using 15 training images per class, which is better than the best published performance by Zhang, et al.

## 1   Introduction

Visual categorization is a difficult task in large part due to the large variation seen between images belonging to the same class. Within one semantic class, there can be a large differences in shape, color, and texture, and objects can be scaled or translated within an image. For some rigid-body objects, appearance changes greatly with viewing angle, and for articulated objects, such as animals, the number of possible configurations can grow exponentially with the degrees of freedom. Furthermore, there is a large number of categories in the world between which humans are able to distinguish. One oft-cited, conservative estimate puts the total at about 30,000 categories [1], and this does not consider the identification problem (e.g. telling faces apart).

One of the more successful tools used in visual classification is a class of patch-based shape and texture features that are invariant or robust to changes in scale, translation, and affine deformations. These include the Gaussian-derivative jet descriptors of [2], SIFT descriptors [3], shape contexts [4], and geometric blur [5]. The basic outline of most discriminative approaches which use these types of features is as follows: (1) given a training image, select a subset of locations or "interest points", (2) for each location, select a patch surrounding it, often elliptical or rectangular in shape, (3) compute a fixed-length feature vector from each patch, usually a summary of edge responses or image gradients. This gives a *set* of fixed-length feature vectors for each training image. (4) Define a function which, given the two sets from two images, returns a value for the distance (or similarity) between the images. Then, (5) use distances between pairs of images as input to a learning algorithm, for example an SVM or nearest neighbor classifier. When given a test image, patches and features are extracted, distances between the test image and training images are computed, and a classification is made.

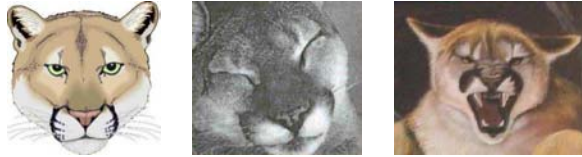

Figure 1: These exemplars are all drawn from the cougar_face category of the Caltech 101 dataset, but we can see a great deal of variation. The image on the left is a clear, color image of a cougar face. As with most cougar_face exemplars, the locations and appearances of the eyes and ears are a strong signal for class membership, as well as the color pattern of the face. Now consider the grayscale center image, where the appearance of the eyes has changed, the ears are no longer visible, and hue is useless. For this image, the markings around the mouth and the texture of the fur become a better signal. The image on the right shows the ears, eyes, and mouth, but due to articulation, the appearance of all have changed again, perhaps representing a common visual sub-category. If we were to limit ourselves to learning one model of relative importance across these features for all images, or even for each category, it could reduce our ability to determine similarity to these exemplars.

In most approaches, machine learning only comes to play in step (5), after the distances or similarities between training images are computed. In this work, we learn the function in step (4) from the training data. This is similar in spirit to the recent body of metric learning work in the machine learning community [6][7][8][9][10]. While these methods have been successfully applied to recognizing digits, there are a couple drawbacks in applying these methods to the general image classification problem. First, they would require representing each image as a fixed-length feature vector. We prefer to use sets of patch-based features, considering both the strong empirical evidence in their favor and the difficulties in capturing invariances in fixed-length feature vectors. Second, these metric-learning algorithms learn one deformation for the entire space of exemplars. To gain an intuition as to why this is a problem, consider Figure 1.

The goal of this paper is to demonstrate that in the setting of visual categorization, it can be useful to determine the relative importance of visual features on a finer scale. In this work, we attack the problem from the other extreme, choosing to learn a *distance function* for *each exemplar*, where each function gives a distance value between its training image, or *focal image*, and any other image. These functions can be learned from either multi-way class labels or relative similarity information in the training data. The distance functions are built on top of elementary distance measures between patch-based features, and our problem is formulated such that we are learning a weighting over the features in each of our training images. This approach has two nice properties: (1) the output of the learning is a quantitative measure of the relative importance of the parts of an image; and (2) the framework allows us to naturally combine and select features of different types.

We learn the weights using a generalization of the constrained optimization formulation proposed by Schultz and Joachims [7] for relative comparison data. Using these local distance functions, we address applications in image browsing, retrieval and classification. In order to perform retrieval and classification, we use an additional learning step that allows us to compare focal images to one another, and an inference procedure based on error-correcting output codes to make a class choice. We show classification results on the Caltech 101 object recognition benchmark, that for some time has been a *de facto* standard for multi-category classification. Our mean recognition rate on this benchmark is 60.3% using only fifteen exemplar images per category, which is an improvement over the best previously published recognition rate in [11].

## 2 Distance Functions and Learning Procedure

In this section we will describe the distance functions and the learning procedure in terms of abstract patch-based image features. Any patch-based features could be used with the framework we present, and we will wait to address our choice of features in Section 3.

If we have $N$ training images, we will be solving $N$ separate learning problems. The training image for which a given learning problem is being solved will be referred to as its *focal image*. Each

problem is trained with a subset of the remaining training images, which we will refer to as the *learning set* for that problem. In the rest of this section we will discuss one such learning problem and focal image, but keep in mind that in the full framework there are $N$ of these.

We define the distance function we are learning to be a combination of *elementary patch-based distances*, each of which are computed between a single patch-based feature in the focal image $\mathcal{F}$ and a *set* of features in a candidate image $\mathcal{I}$, essentially giving us a patch-to-image distance. Any function between a patch feature and a set of features could be used to compute these elementary distances; we will discuss our choice in Section 3. If there are $M$ patches in the focal image, we have $M$ patch-to-image distances to compute between $\mathcal{F}$ and $\mathcal{I}$, and we notate each distance in that set as $d_j^{\mathcal{F}}(\mathcal{I})$, where $j \in [1, M]$, and refer to the vector of these as $\mathbf{d}^{\mathcal{F}}(\mathcal{I})$. The image-to-image distance function $D$ that we learn is a linear combination of these elementary distances. Where $\mathbf{w}^{\mathcal{F}}$ is a vector of weights with a weight corresponding to each patch feature:

$$D(\mathcal{F}, \mathcal{I}) = \sum_{j=1}^{M} w_j^{\mathcal{F}} d_j^{\mathcal{F}}(\mathcal{I}) = \left\langle \mathbf{w}^{\mathcal{F}} \cdot \mathbf{d}^{\mathcal{F}}(\mathcal{I}) \right\rangle \qquad (1)$$

Our goal is to learn this weighting over the features in the focal image. We set up our algorithm to learn from "triplets" of images, each composed of (1) the focal image $\mathcal{F}$, (2) an image labeled "less similar" to $\mathcal{F}$, and (3) an image labeled "more similar" to $\mathcal{F}$. This formulation has been used in other work for its flexibility [7]; it makes it possible to use a relative ranking over images as training input, but also works naturally with multi-class labels by considering exemplars of the same class as $\mathcal{F}$ to be "more similar" than those of another class.

To set up the learning algorithm, we consider one such triplet: $(\mathcal{F}, \mathcal{I}^{\mathrm{d}}, \mathcal{I}^{\mathrm{s}})$, where $\mathcal{I}^{\mathrm{d}}$ and $\mathcal{I}^{\mathrm{s}}$ refer to the dissimilar and similar images, respectively. If we could use our learned distance function for $\mathcal{F}$ to rank these two images relative to one another, we ideally would want $\mathcal{I}^{\mathrm{d}}$ to have a larger value than $\mathcal{I}^{\mathrm{s}}$, i.e. $D(\mathcal{F}, \mathcal{I}^{\mathrm{d}}) > D(\mathcal{F}, \mathcal{I}^{\mathrm{s}})$. Using the formula from the last section, this is equivalent to $\left\langle \mathbf{w}^{\mathcal{F}} \cdot \mathbf{d}^{\mathcal{F}}(\mathcal{I}^{\mathrm{d}}) \right\rangle > \left\langle \mathbf{w}^{\mathcal{F}} \cdot \mathbf{d}^{\mathcal{F}}(\mathcal{I}^{\mathrm{s}}) \right\rangle$. Let $\mathbf{x}_i = \mathbf{d}^{\mathcal{F}}(\mathcal{I}^{\mathrm{d}}) - \mathbf{d}^{\mathcal{F}}(\mathcal{I}^{\mathrm{s}})$, the difference of the two elementary distance vectors for this triplet, now indexed by $i$. Now we can write the condition as $\left\langle \mathbf{w}^{\mathcal{F}} \cdot \mathbf{x}_i \right\rangle > 0$.

For a given focal image, we will construct $T$ of these triplets from our training data (we will discuss how we choose triplets in Section 5.1). Since we will not be able to find one set of weights that meets this condition for all triplets, we use a maximal-margin formulation where we allow slack for triplets that do not meet the condition and try to minimize the total amount of slack allowed. We also increase the desired margin from zero to one, and constrain $\mathbf{w}^{\mathcal{F}}$ to have non-negative elements, which we denote using $\succeq$.[1]

$$\begin{aligned} \arg\min_{\mathbf{w}^{\mathcal{F}}, \boldsymbol{\xi}} \quad & \tfrac{1}{2} \left\| \mathbf{w}^{\mathcal{F}} \right\|^2 + C \sum_{i=1}^{T} \xi_i \\ \text{s.t.} : \quad & \forall (i) \in [1, T] : \left\langle \mathbf{w}^{\mathcal{F}} \cdot \mathbf{x}_i \right\rangle \geq 1 - \xi_i, \xi_i \geq 0 \\ & \mathbf{w}^{\mathcal{F}} \succeq 0 \end{aligned} \qquad (2)$$

We chose the $L_2$ regularization in order to be more robust to outliers and noise. Sparsity is also desirable, and an $L_1$ norm could give more sparse solutions. We do not yet have a direct comparison between the two within this framework.

This optimization is a generalization of that proposed by Schultz and Joachims in [7] for distance metric learning. However, our setting is different from theirs in two ways. First, their triplets do not share the same focal image as they apply their method to learning one metric for all classes and instances. Second, they arrive at their formulation by assuming that (1) each exemplar is represented by a single fixed-length vector, and (2) a $L_2^2$ distance between these vectors is used. This would appear to preclude our use of patch features and more interesting distance measures, but as we show, this is an unnecessary restriction for the optimization. Thus, a contribution of this paper is to show that the algorithm in [7] is more widely applicable than originally presented.

We used a custom solver to find $\mathbf{w}^{\mathcal{F}}$, which runs on the order of one to two seconds for about 2,000 triplets. While it closely resembles the form for support vector machines, it differs in two important ways: (1) we have a primal positivity constraint on $\mathbf{w}^{\mathcal{F}}$, and (2) we do not have a bias term because

we are using the relative relationship between our data vectors. The missing bias term means that, in the dual optimization problem, we do not have a constraint that ties together the dual variables for the margin constraints. Instead, they can be updated separately using an approach similar to the row action method described in [12], followed by a projection of the new $\mathbf{w}^{\mathcal{F}}$ to make it positive. Denoting the dual variables for the margin constraints by $\alpha_i$, we first initialize all $\alpha_i$ to zero, then cycle through the triplets, performing these two steps for the $i$th triplet:

$$\mathbf{w}^{\mathcal{F}} \leftarrow \max\left\{\sum_{i=1}^{T} \alpha_i \mathbf{x}_i, \mathbf{0}\right\}, \quad \alpha_i \leftarrow \min\left\{\max\left\{\frac{1 - \langle \mathbf{w}^{\mathcal{F}} \cdot \mathbf{x}_i \rangle}{\|\mathbf{x}_i\|^2} + \alpha_i, 0\right\}, C\right\}$$

where the first max is element-wise, and the min and max in the second line forces $0 \le \alpha_i \le C$. We stop iterating when all KKT conditions are met, within some precision.

## 3 Visual Features and Elementary Distances

The framework described above allows us to naturally combine different kinds of patch-based features, and we will make use of shape features at two different scales and a rudimentary color feature. Many papers have shown the benefits of using filter-based patch features such as *SIFT* [3] and *geometric blur* [13] for shape- or texture-based object matching and recognition [14][15][13]. We chose to use geometric blur descriptors, which were used by Zhang et al. in [11] in combination with their KNN-SVM method to give the best previously published results on the Caltech 101 image recognition benchmark. Like SIFT, geometric blur features summarize oriented edges within a patch of the image, but are designed to be more robust to affine transformation and differences in the periphery of the patch. In previous work using geometric blur descriptors on the Caltech 101 dataset [13][11], the patches used are centered at 400 or fewer edge points sampled from the image, and features are computed on patches of a fixed scale and orientation. We follow this methodology as well, though one could use an interest point operator to determine location, scale, and orientation from low-level information, as is typically done with SIFT features. We use two different scales of geometric blur features, the same used in separate experiments in [11]. The larger has a patch radius of 70 pixels, and the smaller a patch radius of 42 pixels. Both use four oriented channels and 51 sample points, for a total of 204 dimensions. As is done in [13], we default to normalizing the feature vector so that the $L_2$ norm is equal to one.

Our color features are histograms of eight-pixel radius patches also centered at edge pixels in the image. Any "pixels" in a patch off the edge of the image are counted in a "undefined" bin, and we convert the HSV coordinates of the remaining points to a Cartesian space where the $z$ direction is value and $(x, y)$ is the Cartesian projection of the hue/saturation dimensions. We divide the $(x, y)$ space into an $11 \times 11$ grid, and make three divisions in the $z$ direction. These were the only parameters that we tested with the color features, choosing not to tune the features to the Caltech 101 dataset. We normalize the bins by the total number of pixels in the patch.

Using these features, we can compute elementary patch-to-image distances. If we are computing the distance between the $j$th patch in the focal image to a candidate image $\mathcal{I}$, we find the closest feature of the same type in $\mathcal{I}$ using the $L_2$ distance, and use that $L_2$ distance as the $j$th elementary patch-to-image distance. We only compare features of the same type, so large geometric blur features are not compared to small geometric blur features. In our experiments we have not made use of geometric relationships between features, but this could be incorporated in a manner similar to that in [11] or [16].

## 4 Image Browsing, Retrieval, and Classification

The learned distance functions induce rankings that could naturally be the basis for a browsing application over a closed set of images. Consider a ranking of images with respect to one focal image, as in Figure 2. The user may see this and decide they want more sunflower images. Clicking on the sixth image shown would then take them to the ranking with that sunflower image as the focal image, which contains more sunflower results. In essence, we can allow a user to navigate "image space" by visual similarity.[2]

We also can make use of these distance functions to perform image retrieval: given a new image $\mathcal{Q}$, return a listing of the $N$ training images (or the top $K$) in order of similarity to $\mathcal{Q}$. If given class labels, we would want images ranked high to be in the same class as $\mathcal{Q}$. While we can use the $N$ distance functions to compute the distance from each of the focal images $\mathcal{F}_i$ to $\mathcal{Q}$, these distances are not directly comparable. This is because (1) the weight vectors for each of the focal vectors are not constrained to share any properties other than non-negativity, (2) the number of elementary distance measures and their potential ranges are different for each focal image, and (3) some learned distance functions are simply better than others at characterizing similarity within their class. To address this in cases where we have multi-class labels, we do a second round of training for each focal image where we fit a logistic classifier to the binary (in-class versus out-of-class) training labels and learned distances. Now, given a query image $\mathcal{Q}$, we can compute a probability that the query is in the same class as each of the focal (training) images, and we can use these probabilities to rank the training images relative to one another. The probabilities are on the same scale, and the logistic also helps to penalize poor focal rankings.[3][4]

To classify a query image, we first run the retrieval method above to get the probabilities for each training image. For each class, we sum the probabilities for all training images from that class, and the query is assigned to the class with the largest total. Formally, if $p_j$ is the probability for the $j$th training image $\mathcal{I}_j$, and $\mathcal{C}$ is the set of classes, the chosen class is $\arg\max_{\mathcal{C}} \sum_{j:\mathcal{I}_j \in \mathcal{C}} p_j$. This can be shown to be a relaxation of the Hamming decoding scheme for the error-correcting output codes in [17] in which the number of focal images is the same for each class.

## 5 Caltech101 Experiments

We test our approach on the Caltech101 dataset [18][5]. This dataset has artifacts that make a few classes easy, but many are quite difficult, and due to the important challenges it poses for scalable object recognition, it has up to this point been one of the *de facto* standard benchmarks for multi-class image categorization/object recognition. The dataset contains images from 101 different categories, with the number of images per category ranging from 31 to 800, with a median of about 50 images. We ignore the background class and work in a forced-choice scenario with the 101 object categories, where a query image must be assigned to one of the 101 categories.

We use the same testing methodology and mean recognition reporting described in Grauman et al. [15]: we use varying numbers of training set sizes (given in number of examples per class), and in each training scenario, test with all other images in the Caltech101 dataset, except the BACKGROUND_Google class. Recognition rate per class is computed, then averaged across classes. This normalizes the overall recognition rate so that the performance for categories with a larger number of test images does not skew the mean recognition rate.

### 5.1 Training data

The images are first resized to speed feature computation. The aspect ratio is maintained, but all images are scaled down to be around $200 \times 300$. We computed features for each of these images as described in Section 3. We used up to 400 of each type of feature (two sizes of geometric blur and one color), for a maximum total of 1,200 features per image. For images with few edge points, we computed fewer features so that the features were not overly redundant. After computing elementary distances, we rescale the distances for each focal image and feature to have a standard deviation of 0.1.

For each focal image we choose a set of triplets for training, and since we are learning similarity for the purposes image classification, we use the category labels on the images in the training set: images that have the same label as the focal image are considered "more similar" than all images that are out of class. Note that the training algorithm allows for a more nuanced training set where an image could be more similar with respect to one image and less similar with respect to another, but

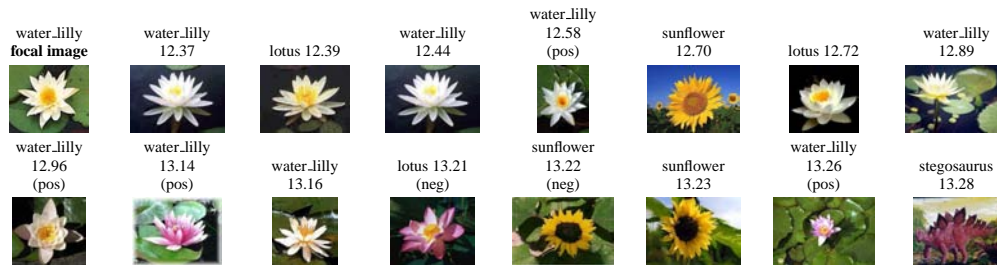

Figure 2: The first 15 images from a ranking induced for the focal image in the upper-left corner, trained with 15 images/category. Each image is shown with its raw distance distance, and only those marked with (pos) or (neg) were in the learning set for this focal image. Full rankings for all experimental runs can be browsed at `http://www.cs.berkeley.edu/~afrome/caltech101/nips2006`.

we are not fully exploiting that in these experiments. Instead of using the full pairwise combination of all in- and out-of-class images, we select triplets using elementary feature distances. Thus, we refer to all the images available for training as the *training set* and the set of images used to train with respect to a given focal image as its *learning set*. We want in our learning set those images that are similar to the focal image according to at least one elementary distance measure. For each of the $M$ elementary patch distance measures, we find the top $K$ closest images. If that group contains both in- and out-of-class images, then we make triplets out of the full bipartite match. If all $K$ images are in-class, then we find the closest out-of-class image according to that distance measure and make $K$ triplets with one out-of-class image and the $K$ similar images. We do the converse if all $K$ images are out of class. In our experiments, we used $K = 5$, and we have not yet performed experiments to determine the effect of the choice of $K$. The final set of triplets for $\mathcal{F}$ is the union of the triplets chosen by the $M$ measures. On average, we used 2,210 triplets per focal image, and mean training time was 1-2 seconds (not including the time to compute the features, elementary distances, or choose the triplets). While we have to solve $N$ of these learning problems, each can be run completely independently, so that for a training set of 1,515 images, we can complete this optimization on a cluster of 50 1GHz computers in about one minute.

## 5.2 Results

We ran a series of experiments using all features, each with a different number of training images per category (either 5, 15, or 30), where we generated 10 independent random splits of the 8,677 images from the 101 categories into training and test sets. We report the average of the mean recognition rates across these splits as well as the standard deviations. We determined the $C$ parameter of the training algorithm using leave-one-out cross-validation on a small random subset of 15 images per category, and our final results are reported using the best value of $C$ found (0.1). In general, however, the method was robust to the choice of $C$, with only changes of about 1% in recognition with an order of magnitude change in $C$ near the maximum. Figure 3 graphs these results with most of the published results for the Caltech 101 dataset.

In the 15 training images per category setting, we also performed recognition experiments on each of our features separately, the combination of the two shape features, and the combination of two shape features with the color features, for a total of five different feature combinations. We performed another round of cross-validation to determine the C value for each feature combination[6]. Recognition in the color-only experiment was the poorest at 6% (0.8% standard deviation)[7] The next best performance was from the bigger geometric blur features with 49.6% ($\pm 1.9\%$), followed by the smaller geometric blur features with 52.1% ($\pm 0.8\%$). Combining the two shape features together, we achieved 58.8% ($\pm 0.8\%$), and with color and shape, reached 60.3% ($\pm 0.7\%$), which

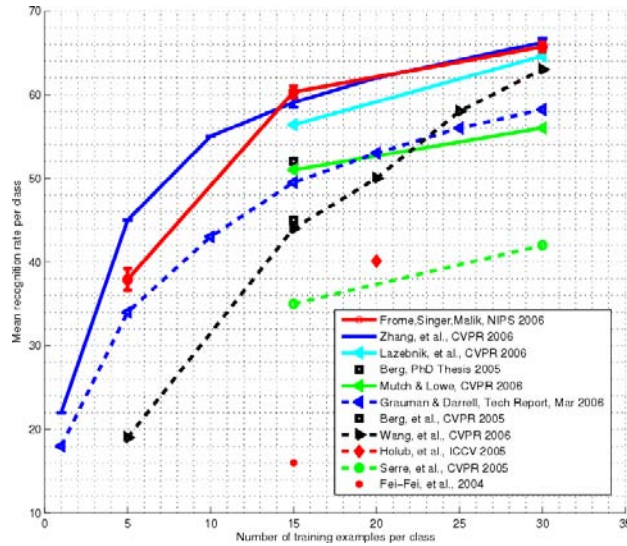

Figure 3: Number of training exemplars versus average recognition rate across classes (based on the graph in [11]). Also shows results from [11], [14], [16], [15], [13], [19], [20], [21], and [18].

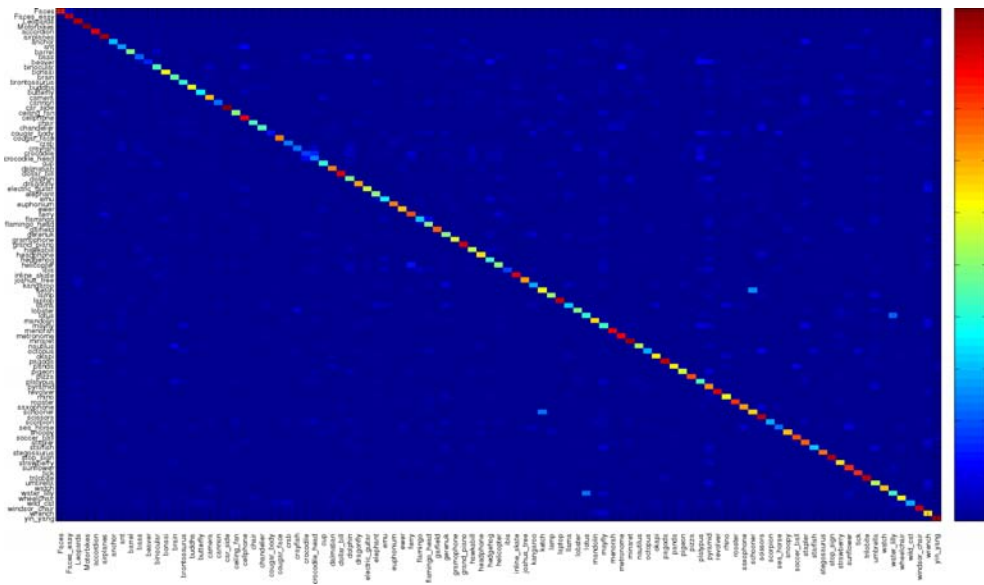

Figure 4: Average confusion matrix for 15 training examples per class, across 10 independent runs. Shown in color using Matlab's jet scale, shown on the right side.

is better than the best previously published performance for 15 training images on the Caltech 101 dataset [11]. Combining shape and color performed better than using the two shape features alone for 52 of the categories, while it degraded performance for 46 of the categories, and did not change performance in the remaining 3. In Figure 4 we show the confusion matrix for combined shape and color using 15 training images per category. The ten worst categories starting with the worst were `cougar_body`, `beaver`, `crocodile`, `ibis`, `bass`, `cannon`, `crayfish`, `sea_horse`, `crab`, and `crocodile_head`, nine of which are animal categories.

Almost all the processing at test time is the computation of the elementary distances between the focal images and the test image. In practice the weight vectors that we learn for our focal images are fairly sparse, with a median of 69% of the elements set to zero after learning, which greatly reduces

the number of feature comparisons performed at test time. We measured that our unoptimized code takes about 300 seconds per test image.[8] After comparisons are computed, we only need to compute linear combinations and compare scores across focal images, which amounts to negligible processing time. This is a benefit of our method compared to the KNN-SVM method of Zhang, et al. [11], which requires the training of a multiclass SVM for every test image, and must perform all feature comparisons.

## Acknowledgements

We would like to thank Hao Zhang and Alex Berg for use of their precomputed geometric blur features, and Hao, Alex, Mike Maire, Adam Kirk, Mark Paskin, and Chuck Rosenberg for many helpful discussions.

## Footnotes

[1]This is based on the intuition that negative weights would mean that larger differences between features could make two images more similar, which is arguably an undesirable effect.

[2]To see a simple demo based on the functions learned for this paper, go to `http://www.cs.berkeley.edu/~afrome/caltech101/nips2006`.

[3]You can also see retrieval rankings with probabilities at the web page.

[4]We experimented with abandoning the max-margin optimization and just training a logistic for each focal image; the results were far worse, perhaps because the logistic was fitting noise in the tails.

[5]Information about the data set, images, and published results can be found at `http://www.vision.caltech.edu/Image_Datasets/Caltech101/Caltech101.html`

[6]For big geometric blur, small geometric blur, both together, and color alone, the values were C=5, 1, 0.5, and 50, respectively.

[7]Only seven categories did better than 33% recognition using only color: `Faces_easy`, `Leopards`, `car_side`, `garfield`, `pizza`, `snoopy`, and `sunflower`. Note that all `car_side` exemplars are in black and white.

[8]To further speed up comparisons, in place of an exact nearest neighbor computation, we could use approximate nearest neighbor algorithms such as locality-sensitive hashing or spill trees.

## References

[1] I. Biederman, "Recognition-by-components: A theory of human image understanding," *Psychological Review*, vol. 94, no. 2, pp. 115–147, 1987.

[2] C. Schmid and R. Mohr, "Combining greyvalue invariants with local constraints for object recognition," in *CVPR*, 1996.

[3] D. Lowe, "Object recognition from local scale-invariant features," in *ICCV*, pp. 1000–1015, Sep 1999.

[4] S. Belongie, J. Malik, and J. Puzicha, "Shape matching and object recognition using shape contexts," *PAMI*, vol. 24, pp. 509–522, April 2002.

[5] A. Berg and J. Malik, "Geometric blur for template matching," in *CVPR*, pp. 607–614, 2001.

[6] E. Xing, A. Ng, and M. Jordan, "Distance metric learning with application to clustering with side-information," in *NIPS*, 2002.

[7] Schutlz and Joachims, "Learning a distance metric from relative comparisons," in *NIPS*, 2003.

[8] S. Shalev-Shwartz, Y. Singer, and A. Ng, "Online and batch learning of pseudo-metrics," in *ICML*, 2004.

[9] K. Q. Weinberger, J. Blitzer, and L. K. Saul, "Distance metric learning for large margin nearest neighbor classification," in *NIPS*, 2005.

[10] A. Globerson and S. Roweis, "Metric learning by collapsing classes," in *NIPS*, 2005.

[11] H. Zhang, A. Berg, M. Maire, and J. Malik, "SVM-KNN: Discriminative Nearset Neighbor Classification for Visual Category Recognition," in *CVPR*, 2006.

[12] Y. Censor and S. A. Zenios, *Parallel Optimization: Theory, Algorithms, and Applications*. Oxford University Press, 1998.

[13] A. Berg, T. Berg, and J. Malik, "Shape matching and object recognition using low distortion correspondence," in *CVPR*, 2005.

[14] S. Lazebnik, C. Schmid, and J. Ponce, "Beyond bags of features: Spatial pyramid matching for recognizing natural scene categories," in *CVPR*, 2006.

[15] K. Grauman and T. Darrell, "Pyramic match kernels: Discriminative classficiation with sets of image features (version 2)," Tech. Rep. MIT_CSAIL_TR_2006-020, MIT, March 2006.

[16] J. Mutch and D. G. Lowe, "Multiclass object recognition with sparse, localized features," in *CVPR*, 2006.

[17] E. L. Allwein, R. E. Schapire, and Y. Singer, "Reducing multiclass to binary: A unifying approach for margin classifiers," *JMLR*, vol. 1, pp. 113–141, 2000.

[18] L. Fei-Fei, R. Fergus, and P. Perona, "Learning generative visual models from few training examples: an incremental bayesian approach testing on 101 object categories.," in *Workshop on Generative-Model Based Vision, CVPR*, 2004.

[19] G. Wang, Y. Zhang, and L. Fei-Fei, "Using dependent regions for object categorization in a generative framework," in *CVPR*, 2006.

[20] A. D. Holub, M. Welling, and P. Perona, "Combining generative models and fisher kernels for object recognition," in *ICCV*, 2005.

[21] T. Serre, L. Wolf, and T. Poggio, "Object recognition with features inspired by visual cortex," in *CVPR*, 2005.

